# Exploratory Feature Extraction in Speech Signals

**Nathan Intrator**
Center for Neural Science
Brown University
Providence, RI 02912

## Abstract

A novel unsupervised neural network for dimensionality reduction which seeks directions emphasizing multimodality is presented, and its connection to exploratory projection pursuit methods is discussed. This leads to a new statistical insight to the synaptic modification equations governing learning in Bienenstock, Cooper, and Munro (BCM) neurons (1982).

The importance of a dimensionality reduction principle based solely on distinguishing features, is demonstrated using a linguistically motivated phoneme recognition experiment, and compared with feature extraction using back-propagation network.

## 1  Introduction

Due to the *curse of dimensionality* (Bellman, 1961) it is desirable to extract features from a high dimensional data space before attempting a classification. How to perform this feature extraction/dimensionality reduction is not that clear. A first simplification is to consider only features defined by linear (or semi-linear) projections of high dimensional data. This class of features is used in projection pursuit methods (see review in Huber, 1985).

Even after this simplification, it is still difficult to characterize what interesting projections are, although it is easy to point at projections that are uninteresting. A statement that has recently been made precise by Diaconis and Freedman (1984) says that for most high-dimensional clouds, most low-dimensional projections are approximately normal. This finding suggests that the important information in the data is conveyed in those directions whose single dimensional projected distribution is far from Gaussian, especially at the center of the distribution. Friedman (1987)

argues that the most computationally attractive measures for deviation from normality (projection indices) are based on polynomial moments. However they very heavily emphasize departure from normality in the tails of the distribution (Huber, 1985). Second order polynomials (measuring the variance - principal components) are not sufficient in characterizing the important features of a distribution (see example in Duda & Hart (1973) p. 212), therefore higher order polynomials are needed. We shall be using the observation that high dimensional clusters translate to multimodal low dimensional projections, and if we are after such structures measuring multimodality defines an interesting projection. In some special cases, where the data is known in advance to be bi-modal, it is relatively straightforward to define a good projection index (Hinton & Nowlan, 1990). When the structure is not known in advance, defining a general multimodal measure of the projected data is not straight forward, and will be discussed in this paper.

There are cases in which it is desirable to make the projection index invariant under certain transformations, and maybe even remove second order structure (see Huber, 1985) for desirable invariant properties of projection indices). In such cases it is possible to make such transformations before hand (Friedman, 1987), and then assume that the data possesses these invariant properties already.

## 2    Feature Extraction using ANN

In this section, the intuitive idea presented above is used to form a statistically plausible objective function whose minimization will be those projections having a single dimensional projected distribution that is far from Gaussian. This is done using a loss function whose expected value leads to the desired projection index. Mathematical details are given in Intrator (1990).

Before presenting this loss function, let us review some necessary notations and assumptions. Consider a neuron with input vector $x = (x_1, \ldots, x_N)$, synaptic weights vector $m = (m_1, \ldots, m_N)$, both in $R^N$, and activity (in the linear region) $c = x \cdot m$. Define the threshold $\Theta_m = E[(x \cdot m)^2]$, and the functions $\hat{\phi}(c, \Theta_m) = c^2 - \frac{2}{3}c\Theta_m$, $\phi(c, \Theta_m) = c^2 - \frac{4}{3}c\Theta_m$. The $\phi$ function has been suggested as a biologically plausible synaptic modification function that explains visual cortical plasticity (Bienenstock, Cooper and Munro, 1982). Note that at this point $c$ represents the linear projection of $x$ onto $m$, and we seek an optimal projection in some sense.

We want to base our projection index on polynomial moments of low order, and to use the fact that bimodal distribution is already interesting, and any additional mode should make the distribution even more interesting. With this in mind, consider the following family of loss functions which depend on the synaptic weight vector and on the input $x$;

$$L_m(x) = -\mu \int_{\Theta_m}^{(x \cdot m)} \hat{\phi}(s, \Theta_m)ds = -\frac{\mu}{3}\{(x \cdot m)^3 - E[(x \cdot m)^2](x \cdot m)^2\}.$$

The motivation for this loss function can be seen in the following graph, which represents the $\phi$ function and the associated loss function $L_m(x)$. For simplicity the loss for a fixed threshold $\Theta_m$ and synaptic vector $m$ can be written as $L_m(c) = -\frac{\mu}{3}c^2(c - \Theta_m)$, where $c = (x \cdot m)$.

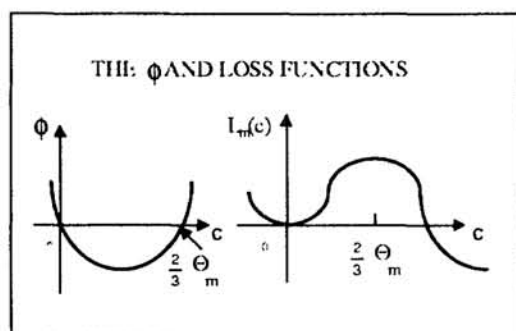

Figure 1: The function $\phi$ and the loss functions for a fixed $m$ and $\Theta_m$.

The graph of the loss function shows that for any fixed $m$ and $\Theta_m$, the loss is small for a given input $x$, when either $(x \cdot m)$ is close to zero, or when $(x \cdot m)$ is larger than $\frac{4}{3}\Theta_m$. Moreover, the loss function remains negative for $(x \cdot m) > \frac{4}{3}\Theta_m$, therefore, any kind of distribution at the right hand side of $\frac{4}{3}\Theta_m$ is possible, and the preferred ones are those which are concentrated further away from $\frac{4}{3}\Theta_m$.

We must still show why it is not possible that a minimizer of the average loss will be such that all the mass of the distribution will be concentrated in one of the regions. Roughly speaking, this can not happen because the threshold $\Theta_m$ is dynamic and depends on the projections in a nonlinear way, namely, $\Theta_m = E(x \cdot m)^2$. This implies that $\Theta_m$ will always move itself to a stable point such that the distribution will not be concentrated at only one of its sides. This yields that the part of the distribution for $c < \frac{4}{3}\Theta_m$ has a high loss, making those distributions in which the distribution for $c < \frac{4}{3}\Theta_m$ has its mode at zero more plausible.

The risk (expected value of the loss) is given by:

$$R_m = -\frac{\mu}{3} \left\{ E[(x \cdot m)^3] - E^2[(x \cdot m)^2] \right\}.$$

Since the risk is continuously differentiable, its minimization can be achieved via a gradient descent method with respect to $m$, namely:

$$\frac{dm_i}{dt} = -\frac{\partial}{\partial m_i} R_m = \mu \, E[\phi(x \cdot m, \Theta_m)x_i].$$

The resulting differential equations suggest a modified version of the law governing synaptic weight modification in the BCM theory for learning and memory (Bienenstock, Cooper and Munro, 1982). This theory was presented to account for various experimental results in visual cortical plasticity. The biological relevance of the theory has been extensively studied (Soul et al., 1986; Bear et al., 1987; Cooper et al., 1987; Bear et al., 1988), and it was shown that the theory is in agreement with the classical deprivation experiments (Clothioux et al., 1990).

The fact that the distribution has part of its mass on both sides of $\frac{4}{3}\Theta_m$ makes this loss a plausible projection index that seeks multimodalities. However, we still need

to reduce the sensitivity of the projection index to outliers, and for full generality, allow any projected distribution to be shifted so that the part of the distribution that satisfies $c < \frac{4}{3}\Theta_m$ will have its mode at zero. The over-sensitivity to outliers is addressed by considering a nonlinear neuron in which the neuron's activity is defined to be $c = \sigma(x \cdot m)$, where $\sigma$ usually represents a smooth sigmoidal function. A more general definition that would allow symmetry breaking of the projected distributions, will provide solution to the second problem raised above, and is still consistent with the statistical formulation, is $c = \sigma(x \cdot m - \alpha)$, for an arbitrary threshold $\alpha$ which can be found by using gradient descent as well. For the nonlinear neuron, $\Theta_m$ is defined to be $\Theta_m = E[\sigma^2(x \cdot m)]$.

Based on this formulation, a network of $Q$ identical nodes may be constructed. All the neurons in this network receive the same input and inhibit each other, so as to extract several features in parallel. A similar network has been studied in the context of mean field theory by Scofield and Cooper (1985). The activity of neuron $k$ in the network is defined as $c_k = \sigma(x \cdot m_k - \alpha_k)$, where $m_k$ is the synaptic weight vector of neuron $k$, and $\alpha_k$ is its threshold. The *inhibited* activity and threshold of the $k$'th neuron are given by $\tilde{c}_k = c_k - \eta \sum_{j \neq k} c_j, \quad \tilde{\Theta}_m^k = E[\tilde{c}_k^2]$.

We omit the derivation of the synaptic modification equations which is similar to the one for a single neuron, and present only the resulting modification equations for a synaptic vector $m_k$ in a lateral inhibition network of nonlinear neurons:

$$\dot{m}_k = -\mu \; E\{\phi(\tilde{c}_k, \tilde{\Theta}_m^k)\Big(\sigma'(c_k) - \eta \sum_{j \neq k} \sigma'(c_j)\Big)x\}.$$

The lateral inhibition network performs a direct search of $Q$-dimensional projections together, and therefore may find a richer structure that a stepwise approach may miss, e.g. see example 14.1 Huber (1985).

## 3    Comparison with other feature extraction methods

When dealing with a classification problem, the interesting features are those that distinguish between classes. The network presented above has been shown to seek multimodality in the projected distributions, which translates to clusters in the original space, and therefore to find those directions that make a distinction between different sets in the training data.

In this section we compare classification performance of a network that performs dimensionality reduction (before the classification) based upon multimodality, and a network that performs dimensionality reduction based upon minimization of misclassification error (using back-propagation with MSE criterion). This is done using a phoneme classification experiment whose linguistic motivation is described below. In the latter we regard the hidden units representation as a new reduced feature representation of the input space. Classification on the new feature space was done using back-propagation[1]

Consider the six stop consonants [p,k,t,b,g,d], which have been a subject of recent research in evaluating neural networks for phoneme recognition (see review in Lippmann, 1989). According to phonetic feature theory, these stops posses several common features, but only two distinguishing phonetic features, place of articulation and voicing (see Blumstein & Lieberman 1984, for a review and related references on phonetic feature theory). This theory suggests an experiment in which features extracted from unvoiced stops can be used to distinguish place of articulation in voiced stops as well. It is of interest if these features can be found from a single speaker, how sensitive they are to voicing and whether they are speaker invariant.

The speech data consists of 20 consecutive time windows of 32msec with 30msec overlap, aligned to the beginning of the burst. In each time window, a set of 22 energy levels is computed. These energy levels correspond to Zwicker critical band filters (Zwicker, 1961). The consonant-vowel (CV) pairs were pronounced in isolation by native American speakers (two male BSS and LTN, and one female JES.) Additional details on biological motivation for the preprocessing, and linguistic motivation related to child language acquisition can be found in Seebach (1990), and Seebach and Intrator (1991). An average (over 25 tokens) of the six stop consonants followed by the vowel [a] is presented in Figure 2. All the images are smoothened using a moving average. One can see some similarities between the voiced and unvoiced stops especially in the upper left corner of the image (high frequencies beginning of the burst) and the radical difference between them in the low frequencies.

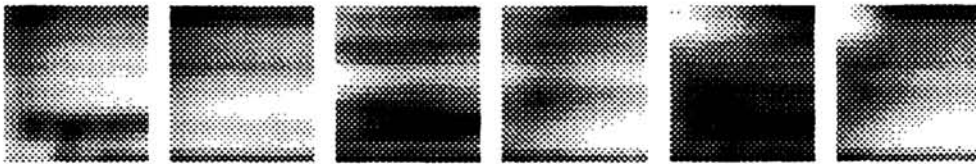

Figure 2: An average of the six stop consonants followed by the vowel [a]. Their order from left to right [pa] [ba] [ka] [ga] [ta] [da]. Time increases from the burst release on the X axis, and frequency increases on the Y axis.

In the experiments reported here, 5 features were extracted from the 440 dimension original space. Although the dimensionality reduction methods were trained only with the unvoiced tokens of a single speaker, the classifier was trained on (5 dimensional) voiced and unvoiced data from the other speakers as well.

The classification results, which are summarized in table 1, show that the back-propagation network does well in finding structure useful for classification of the trained data, but this structure is more sensitive to voicing. Classification results using a BCM network suggest that, for this specific task, structure that is less sensitive to voicing can be extracted, even though voicing has significant effects on the speech signal itself. The results also suggest that these features are more speaker invariant.

| Place of Articulation Classification (B-P) | | |
|---|---|---|
| | B-P | BCM |
| BSS /p,k,t/ | 100 | 100 |
| BSS /b,g,d/ | 83.4 | 94.7 |
| LTN /p,k,t/ | 95.6 | 97.7 |
| LTN /b,g,d/ | 78.3 | 93.2 |
| JES (Both) | 88.0 | 99.4 |

Table 1: Percentage of correct classification of place of articulation in voiced and unvoiced stops.

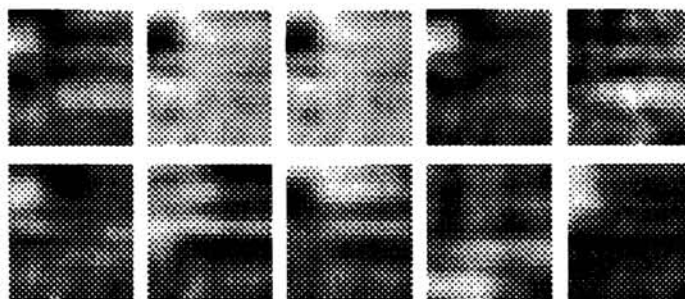

Figure 3 : Synaptic weight images of the 5 hidden units of back-propagation (top), and by the 5 BCM neurons (bottom).

The difference in performance between the two feature extractors may be partially explained by looking at the synaptic weight vectors (images) extracted by both method: For the back-propagation feature extraction it can be seen that although 5 units were used, fewer number of features were extracted. One of the main distinction between the unvoiced stops in the training set is the high frequency burst at the beginning of the consonant (the upper left corner). The back-propagation method concentrated mainly on this feature, probably because it is sufficient to base the recognition of the training set on this feature, and the fact that training stops when misclassification error falls to zero. On the other hand, the BCM method does not try to reduce the misclassificaion error and is able to find a richer, linguistically meaningful structure, containing burst locations and format tracking of the three different stops that allowed a better generalization to other speakers and to voiced stops.

The network and its training paradigm present a different approach to speaker independent speech recognition. In this approach the speaker variability problem is addressed by training a network that concentrates mainly on the distinguishing features of a single speaker, as opposed to training a network that concentrates on both the distinguishing and common features, on multi-speaker data.

## Acknowledgements

I wish to thank Leon N Cooper for suggesting the problem and for providing many helpful hints and insights. Geoff Hinton made invaluable comments. The application of BCM to speech is discussed in more detail in Seebach (1990) and in a

forthcoming article (Seebach and Intrator, 1991). Research was supported by the National Science Foundation, the Army Research Office, and the Office of Naval Research.

## Footnotes

[1]See Intrator (1990) for comparison with principal components feature extraction and with k-NN as a classifier

## References

Bellman, R. E. (1961) Adaptive Control Processes, Princeton, NJ, Princeton University Press.

Bienenstock, E. L., L. N Cooper, and P.W. Munro (1982) Theory for the development of neuron selectivity: orientation specificity and binocular interaction in visual cortex. *J.Neurosci.* 2:32-48

Bear, M. F., L. N Cooper, and F. F. Ebner (1987) A Physiological Basis for a Theory of Synapse Modification. *Science* 237:42-48

Diaconis, P, and D. Freedman (1984) Asymptotics of Graphical Projection Pursuit. *The Annals of Statistics*, 12 793-815.

Friedman, J. H. (1987) Exploratory Projection Pursuit. *Journal of the American Statistical Association* 82-397:249-266

Hinton, G. E. and S. J. Nowlan (1990) The bootstrap Widrow-Hoff rule as a cluster-formation algorithm. *Neural Computation.*

Huber P. J. (1985) Projection Pursuit. *The Annal. of Stat.* 13:435-475

Intrator N. (1990) A Neural Network For Feature Extraction. In D. S. Touretzky (ed.), *Advances in Neural Information Processing Systems 2.* San Mateo, CA: Morgan Kaufmann.

Lippmann, R. P. (1989) Review of Neural Networks for Speech Recognition. *Neural Computation* 1, 1-38.

Reilly, D. L., C.L. Scofield, L. N Cooper and C. Elbaum (1988) GENSEP: a multiple neural network with modifiable network topology. *INNS Conference on Neural Networks.*

Saul, A. and E. E. Clothiaux, 1986) Modeling and Simulation II: Simulation of a Model for Development of Visual Cortical specificity. *J. of Electrophysiological Techniques,* 13:279-306

Scofield, C. L. and L. N Cooper (1985) Development and properties of neural networks. *Contemp. Phys.* 26:125-145

Seebach, B. S. (1990) Evidence for the Development of Phonetic Property Detectors in a Neural Net without Innate Knowledge of Linguistic Structure. Ph.D. Dissertation Brown University.

Duda R. O. and P. E. Hart (1973) *Pattern classification and scene analysis* John Wiley, New York

Zwicker E. (1961) Subdivision of the audible frequency range into critical bands (Frequenzgruppen) *Journal of the Acoustical Society of America* 33:248